# Broadband Direction-Of-Arrival Estimation Based On Second Order Statistics

Justinian Rosca     Joseph Ó Ruanaidh     Alexander Jourjine     Scott Rickard

{rosca,oruanaidh,jourjine,rickard}@scr.siemens.com

Siemens Corporate Research, Inc.
755 College Rd E
Princeton, NJ 08540

## Abstract

$N$ wideband sources recorded using $N$ closely spaced receivers can feasibly be separated based only on second order statistics when using a physical model of the mixing process. In this case we show that the parameter estimation problem can be essentially reduced to considering directions of arrival and attenuations of each signal. The paper presents two demixing methods operating in the time and frequency domain and experimentally shows that it is always possible to demix signals arriving at different angles. Moreover, one can use spatial cues to solve the channel selection problem and a post-processing Wiener filter to ameliorate the artifacts caused by demixing.

## 1 Introduction

Blind source separation (BSS) is capable of dramatic results when used to separate mixtures of independent signals. The method relies on simultaneous recordings of signals from two or more input sensors and separates the original sources purely on the basis of statistical independence between them. Unfortunately, BSS literature is primarily concerned with the idealistic instantaneous mixing model.

In this paper, we formulate a low dimensional and fast solution to the problem of separating two signals from a mixture recorded using two closely spaced receivers. Using a physical model of the mixing process reduces the complexity of the model and allows one to identify and to invert the mixing process using second order statistics only.

We describe the theoretical basis of the new approach, and then focus on two algorithms, which were implemented and successfully applied to extensive sets of real-world data. In essence, our separation architecture is a system of adaptive directional receivers designed using the principles of BSS. The method bears resemblance to methods in beamforming [8] in that it works by spatial filtering. Array processing techniques [2] reduce noise by separating signal space from noise space, which necessitates more receivers than emitters. The main differences are that standard beamforming and array processing techniques [8, 2] are generally strictly concerned with processing directional narrowband signals. The difference with BSS [7, 6] is that our approach is model-based and therefore the elements of the mixing matrix are highly constrained: a feature that aids in the robust and reliable identification of the mixing process.

The layout of the paper is as follows. Sections 2 and 3 describe the theoretical foundation of the separation method that was pursued. Section 4 presents algorithms that were developed and experimental results. Finally we summarize and conclude this work.

## 2  Theoretical foundation for the BSS solution

As a first approximation to the general multi-path model, we use the delay-mixing model. In this model, only direct path signal components are considered. Signal components from one source arrive with a fractional delay between the time of arrivals at two receivers. By fractional delays, we mean that delays between receivers are not generally integer multiples of the sampling period. The delay depends on the position of the source with respect to the receiver axis and the distance between receivers. Our BSS algorithms demix by compensating for the fractional delays. This, in effect, is a form of adaptive beamforming with directional notches being placed in the direction of sources of interference [8]. A more detailed account of the analytical structure of the solutions can be found in [1].

Below we address the case of two inputs and two outputs but there is no reason why the discussion cannot be generalized to multiple inputs and multiple outputs. Assume a linear mixture of two sources, where source amplitude drops off in proportion to distance:

$$x_i(t) = \frac{1}{R_{i1}} s_1(t - \frac{R_{i1}}{c}) + \frac{1}{R_{i2}} s_2(t - \frac{R_{i2}}{c}) \tag{1}$$

$j = 1, 2$, where $c$ is the speed of wave propagation, and $R_{ij}$ indicates the distance from receiver $i$ to source $j$. This describes signal propagation through a uniform non-dispersive medium. In the Fourier domain, Equation 1 results in a *mixing matrix $A(\omega)$* given by:

$$A(\omega) = \begin{bmatrix} \frac{1}{R_{11}} e^{-j\omega \frac{R_{11}}{c}} & \frac{1}{R_{12}} e^{-j\omega \frac{R_{12}}{c}} \\ \frac{1}{R_{21}} e^{-j\omega \frac{R_{21}}{c}} & \frac{1}{R_{22}} e^{-j\omega \frac{R_{22}}{c}} \end{bmatrix} \tag{2}$$

It is important to note that the columns can be scaled arbitrarily without affecting separation of sources because rescaling is absorbed into the sources. This implies that row scaling in the demixing matrix (the inverse of $A(\omega)$) is arbitrary.

Using the Cosine Rule, $R_{ij}$ can be expressed in terms of the distance $R_j$ of source $j$ to the midpoint between two receivers, the direction of arrival of source $j$, and the distance between receivers, $d$, as follows:

$$R_{ij} = \left[ R_j^2 + \left( \frac{d}{2} \right)^2 + 2(-1)^i \left( \frac{d}{2} \right) R_j \cos \theta_j \right]^{\frac{1}{2}} \tag{3}$$

Expanding the right term above using the binomial expansion and preserving only zeroth and first order terms, we can express distance from the receivers to the sources as:

$$R_{ij} = \left( R_j + \frac{d^2}{8R_j} \right) + (-1)^i \left( \frac{d}{2} \right) \cos \theta_j \tag{4}$$

This approximation is valid within a 5% relative error when $d \leq \frac{R_j}{2}$. With the substitution for $R_{ij}$ and with the redefinition of source $j$ to include the delay due to the term within brackets in Equation 4 divided by $c$, Equation 1 becomes:

$$x_i(t) = \sum_j \frac{1}{R_{ij}} \cdot s_j \left( t + (-1)^i \cdot \left( \frac{d}{2c} \right) \cdot \cos \theta_j \right), i = 1, 2 \tag{5}$$

In the Fourier domain, equation 5 results in the simplification to the mixing matrix $A(\omega)$:

$$A(\omega) = \begin{bmatrix} \frac{1}{R_{11}} \cdot e^{-j\omega \delta_1} & \frac{1}{R_{12}} \cdot e^{-j\omega \delta_2} \\ \frac{1}{R_{21}} \cdot e^{j\omega \delta_1} & \frac{1}{R_{22}} \cdot e^{j\omega \delta_2} \end{bmatrix} \tag{6}$$

Here phases are functions of the directions of arrival $\theta_j$ (defined with respect to the midpoint between receivers), the distance between receivers $d$, and the speed of propagation $c$: $\delta_i = \frac{d}{2c}\cos\theta_i, i = 1, 2$. $R_{ij}$ are unknown, but we can again redefine sources so diagonal elements are unity:

$$A(\omega) = \begin{bmatrix} e^{-j\omega\delta_1} & c_1.e^{-j\omega\delta_2} \\ c_2.e^{j\omega\delta_1} & e^{j\omega\delta_2} \end{bmatrix} \tag{7}$$

where $c_1$, $c_2$ are two positive real numbers. In wireless communications sources are typically distant compared to antenna distance. For distant sources and a well matched pair of receivers $c_1 \approx c_2 \approx 1$. Equation 7 describes the mixing matrix for the delay model in the frequency domain, in terms of four parameters, $\delta_1, \delta_2, c_1, c_2$.

The corresponding ideal demixing matrix $W(\omega)$, for each frequency $\omega$, is given by:

$$W(\omega) = [A(\omega)]^{-1} = \frac{1}{\det A(\omega)} \begin{bmatrix} e^{j\omega\delta_2} & -c_1.e^{-j\omega\delta_2} \\ -c_2.e^{j\omega\delta_1} & e^{-j\omega\delta_1} \end{bmatrix} \tag{8}$$

The outputs, estimating the sources, are:

$$\begin{bmatrix} z_1(\omega) \\ z_2(\omega) \end{bmatrix} = W(\omega) \begin{bmatrix} x_1(\omega) \\ x_2(\omega) \end{bmatrix} = \frac{1}{\det A(\omega)} \begin{bmatrix} e^{j\omega\delta_2} & -c_1 e^{-j\omega\delta_2} \\ -c_2 e^{j\omega\delta_1} & e^{-j\omega\delta_1} \end{bmatrix} \begin{bmatrix} x_1(\omega) \\ x_2(\omega) \end{bmatrix} \tag{9}$$

Making the transition back to the time domain results in the following estimate of the outputs:

$$\begin{bmatrix} z_1(t) \\ z_2(t) \end{bmatrix} = h(t, \delta_1, \delta_2, c_1, c_2) \otimes \begin{bmatrix} x_1(t + \delta_2) - c_1\, x_2(t - \delta_2) \\ -c_2\, x_1(t + \delta_1) + x_2(t - \delta_1) \end{bmatrix} \tag{10}$$

where $\otimes$ is convolution, and

$$h(t, \delta_1, \delta_2, c_1, c_2) = \frac{1}{2\pi} \int e^{j\omega t} H(\omega, \delta_1, \delta_2, c_1, c_2) d\omega \tag{11}$$

$$H(\omega, \delta_1, \delta_2, c_1, c_2) = \frac{1}{\det A(\omega)} = \frac{1}{e^{j\omega(\delta_2 - \delta_1)} - c_1\, c_2\, e^{-j\omega(\delta_2 - \delta_1)}}$$

Formulae 9 and 10 form the basis for two algorithms to be described next, in the time domain and the frequency domains. The algorithms have the role of determining the four unknown parameters. Note that the filter corresponding to $H(\omega, \delta_1, \delta_2, c_1, c_2)$ should be applied to the output estimates in order to map back to the original inputs.

## 3 Delay and attenuation compensation algorithms

The estimation of the four unknown parameters $\delta_1$, $\delta_2$, $c_1$, $c_2$ can be carried out based on second order criteria that impose the constraint that outputs are decorrelated ([9, 4, 6, 5]).

### 3.1 Time and frequency domain approaches

The time domain algorithm is based on the idea of imposing the decorrelation constraint $\langle z_1(t), z_2(t)\rangle = 0$ between the estimates of the outputs, as a function of the delays $D_1$ and $D_2$ and scalar coefficients $c_1$ and $c_2$. This is equivalent to the following criterion:

$$\{\hat{D}_1, \hat{D}_2, \hat{c}_1, \hat{c}_2\} = \mathrm{argmin}\{F(D_1, D_2, c_1, c_2)\} \tag{12}$$

where $F(.)$ measures the cross-correlations between the signals given below, representing filtered versions of the differences of fractionally delayed measurements:

$$z_1(t) = h(t, D_1, D_2, c_1, c_2) \otimes \left(x_1(t + D_2) - c_1 x_2(t)\right) \tag{13}$$
$$z_2(t) = h(t, D_1, D_2, c_1, c_2) \otimes \left(c_2 x_1(t + D_2) - x_2(t)\right)$$
$$F(D_1, D_2, c_1, c_2) = \langle z_1(t), z_2(t) \rangle$$

In the frequency domain, the cross-correlation of the inputs is expressed as follows:

$$R^X(\omega) = A(\omega) R^S(\omega) A^H(\omega) \tag{14}$$

The mixing matrix in the frequency domain has the form given in Equation 7. Inverting this cross correlation equation yields four equations that are written in matrix form as:

$$R^S(\omega) = A^{-1}(\omega) R^X(\omega) A^{-H}(\omega) \tag{15}$$

Source orthogonality implies that the off-diagonal terms in the covariance matrix must be zero:

$$R^S_{12}(\omega) = 0 \tag{16}$$
$$R^S_{21}(\omega) = 0$$

For far field conditions (i.e. the distance between the receivers is much less than the distance from sources) one obtains the following equations:

$$R^S_{12}(\omega) = \bar{c}_1 \frac{a}{b} R^X_{11}(\omega) - c_2 \frac{b}{a} R^X_{22}(\omega) - \bar{c}_1 c_2 ab R^X_{21}(\omega) - \frac{1}{ab} R^X_{12}(\omega) = 0 \tag{17}$$
$$R^S_{21}(\omega) = c_1 \frac{b}{a} R^X_{11}(\omega) - \bar{c}_2 \frac{a}{b} R^X_{22}(\omega) - ab R^X_{21}(\omega) - \frac{c_1 \bar{c}_2}{ab} R^X_{12}(\omega) = 0$$

The terms $a = e^{-j\omega\delta_1}$ and $b = e^{-j\omega\delta_2}$ are functions of the time delays. Note that there is a pair of equations of this kind for each frequency. In practice, the unknowns should be estimated from data at all available frequencies to obtain a robust estimate.

## 3.2 Channel selection

Up to this point, there was no guarantee that estimated parameters would ensure source separation in some specific order. We could not decide a priori whether estimated parameters for the first output channel correspond to the first or second source. However, the dependence of the phase delays on the angles of arrival suggests a way to break the permutation symmetry in source estimation, that is to decide precisely which estimate to present on the first channel (and henceforth on the second channel as well).

The core idea is that directionality and spatial cues provide the information required to break the symmetry. The criterion we use is to sort sources in order of increasing delay. Note that the correspondence between delays and sources is unique when sources are not symmetrical with respect to the receiver axis. When sources are symmetric there is no way of distinguishing between their positions because the cosine of the angles of arrival, and hence the delay, is invariant to the sign of the angle.

## 4 Experimental results

A robust implementation of criterion 12 averages cross-correlations over a number of windows, of given size. More precisely F is defined as follows:

$$F(\delta_1, \delta_2) = \sum_{\text{Blocks}} |\langle z_1(t), z_2(t) \rangle|^q \tag{18}$$

Normally $q = 1$ to obtain a robust estimate. Ngo and Bhadkamkar [5] suggest a similar criterion using $q = 2$ without making use of the determinant of the mixing matrix.

After taking into account all terms from Equation 18, including the determinant of the mixing matrix A, we obtain the function to be used for parameter estimation in the frequency domain:

$$F(\delta_1, \delta_2) = \sum_\omega \frac{1}{\{\det A\}^2 + \eta} \cdot \left| \frac{a}{b} R_{11}^X(\omega) - \frac{b}{a} R_{22}^X(\omega) - ab R_{21}^X(\omega) - \frac{1}{ab} R_{12}^X(\omega) \right|^q \quad (19)$$

where $\eta$ is a (Wiener Filter-like) constant that helps prevent singularities and $q$ is normally set to one.

Computing the separated sources using only time differences leads to highpass filtered outputs. In order to implement exactly the theoretical demixing procedure presented one has to divide by the determinant of the mixing matrix. Obviously one could filter using the inverse of the determinant to obtain optimal results. This can be implemented in the form of a Wiener filter. The Wiener filter requires knowledge both of the signal and noise power spectral densities. This information is not available to us but a reasonable approximation is to assume that the (wideband) sources have a flat spectral density and the noise corrupting the mixtures is white. In this case, the Wiener Filter becomes:

$$H(\omega) = \left( \frac{\{\det A(\omega)\}^2}{\{\det A(\omega)\}^2 + \eta} \right) \frac{1}{\det A(\omega)} \quad (20)$$

where the parameter $\eta$ has been empirically set to the variance of the mixture. Applying this choice of filter usually dramatically improves the quality of the separated outputs.

The technique of postprocessing using the determinant of the mixing matrix is perfectly general and applies equally well to demixtures computed using matrices of FIR filters. The quality of the result depends primarily on the care with which the inverse filter is implemented. It also depends on the accuracy of the estimate for the mixing parameters. One should avoid using the Wiener filter for near-degenerate mixtures.

The proof of concept for the theory outlined above was obtained using speech signals which if anything pose a greater challenge to separation algorithms because of the correlation structure of speech. Two kinds of data are considered in this paper: synthetic direct propagation delay data and synthetic multi-path data. Data can be characterized along two dimensions of difficulty: synthetic vs. real-world, and direct path vs. multi-path. Combinations along these dimensions represented the main type of data we used.

The value of distance between receivers dictates the order of delays that can appear due to direct path propagation, which is used by the demixing algorithms. Data was generated synthetically employing fractional delays corresponding to the various positions of the sources [3].

We modeled multi-path by taking into account the decay in signal amplitude due to propagation distance as well as the absorption of waves. Only the direct path and one additional path were considered.

The algorithms developed proved successful for separation of two voices from direct path mixtures, even where the sources had very similar spectral power characteristics, and for separation of one source for multi-path mixtures. Moreover, outputs were free from artifacts and were obtained with modest computational requirements.

Figure 1 presents mean separation results of the first and second channels, which correspond to the first and second sources, for various synthetic data sets. Separation depends on the angles of arrival. Plots show no separation in the degenerate case of equal or closeby angles of arrival, but more than 10dB mean separation in the anechoic case and 5dB in the multi-path case.

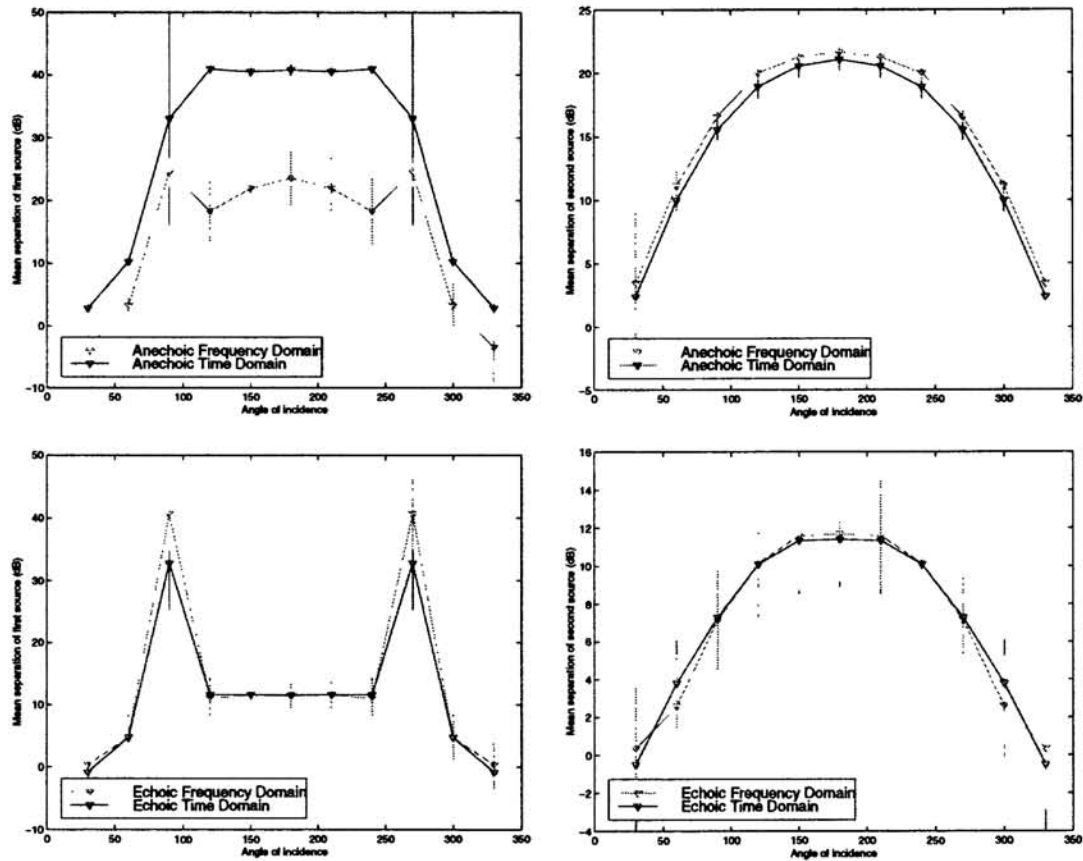

Figure 1: Two sources were positioned at a relatively large distance from a pair of closely spaced receivers. The first source was always placed at zero degrees whilst the second source was moved uniformly from 30 to 330 degrees in steps of 30 degrees. The above shows mean separation and standard deviation error bars of first and second sources for six synthetic delay mixtures or synthetic multi-path data mixtures using the time and frequency domain algorithms.

## 5   Conclusions

The present source separation approach is based on minimization of cross-correlations of the estimated sources, in the time or frequency domains, when using a delay model and explicitly employing dirrection of arrival. The great advantage of this approach is that it reduces source separation to a decorrelation problem, which is theoretically solved by a system of equations. Although the delay model used generates essentially anechoic time delay algorithms, the results of this work show systematic improvements even when the algorithms are applied to real multi-path data. In all cases separation improvement is robust with respect to the power ratios of sources.

## Acknowledgments

We thank Radu Balan and Frans Coetzee for useful discussions and proofreading various versions of this document and our collaborators within Siemens for providing extensive data for testing.

# References

[1] A. Jourjine, S. Rickard, J. Ó Ruanaidh, and J. Rosca. Demixing of anechoic time delay mixtures using second order statistics. Technical Report SCR-99-TR-657, Siemens Corporate Research, 755 College Road East, Princeton, New Jersey, 1999.

[2] Hamid Krim and Mats Viberg. Two decades of array signal processing research. *IEEE Signal Processing Magazine*, 13(4), 1996.

[3] Tim Laakso, Vesa Valimaki, Matti Karjalainen, and Unto Laine. Splitting the unit delay. *IEEE Signal Processing Magazine*, pages 30–60, 1996.

[4] L. Molgedey and H.G. Schuster. Separation of a mixture of independent signals using time delayed correlations. *Phys.Rev.Lett.*, 72(23):3634–3637, July 1994.

[5] T. J. Ngo and N.A. Bhadkamkar. Adaptive blind separation of audio sources by a physically compact device using second order statistics. In *First International Workshop on ICA and BSS*, pages 257–260, Aussois, France, January 1999.

[6] Lucas Parra, Clay Spence, and Bert De Vries. Convolutive blind source separation based on multiple decorrelation. In *NNSP98*, 1988.

[7] K. Torkolla. Blind separation for audio signals: Are we there yet? In *First International Workshop on Independent component analysis and blind source separation*, pages 239–244, Aussois, France, January 1999.

[8] V. Van Veen and Kevin M. Buckley. Beamforming: A versatile approach to spatial filtering. *IEEE ASSP Magazine*, 5(2), 1988.

[9] E. Weinstein, M. Feder, and A. Oppenheim. Multi-channel signal separation by decorrelation. *IEEE Trans. on Speech and Audio Processing*, 1(4):405–413, 1993.